# A competitive modular connectionist architecture

**Robert A. Jacobs** and **Michael I. Jordan**
Department of Brain & Cognitive Sciences
Massachusetts Institute of Technology
Cambridge, MA 02139

## Abstract

We describe a multi–network, or modular, connectionist architecture that captures that fact that many tasks have structure at a level of granularity intermediate to that assumed by local and global function approximation schemes. The main innovation of the architecture is that it combines associative and competitive learning in order to learn task decompositions. A task decomposition is discovered by forcing the networks comprising the architecture to compete to learn the training patterns. As a result of the competition, different networks learn different training patterns and, thus, learn to partition the input space. The performance of the architecture on a "what" and "where" vision task and on a multi–payload robotics task are presented.

## 1 INTRODUCTION

A dichotomy has arisen in recent years in the literature on nonlinear network learning rules between *local* approximation of functions and *global* approximation of functions. Local approximation, as exemplified by lookup tables, nearest–neighbor algorithms, and networks with units having local receptive fields, has the advantage of requiring relatively few learning trials and tends to yield interpretable representations. Global approximation, as exemplified by polynomial regression and fully–connected networks with sigmoidal units, has the advantage of requiring less storage capacity than local approximators and may yield superior generalization.

In this paper, we report a multi–network, or modular, connectionist architecture that captures the fact that many tasks have structure at a level of granularity intermediate to that assumed by local and global approximation schemes. It does so

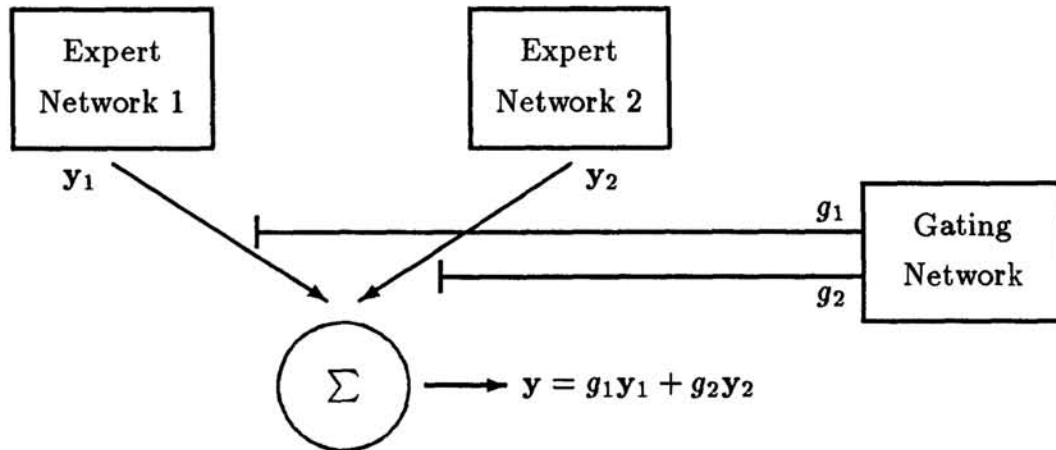

Figure 1: A Modular Connectionist Architecture

by combining the desirable features of the approaches embodied by these disparate approximation schemes. In particular, it uses different networks to learn training patterns from different regions of the input space. Each network can itself be a local or global approximator for a particular region of the space.

## 2   A MODULAR CONNECTIONIST ARCHITECTURE

The technical issues addressed by the modular architecture are twofold: (a) detecting that different training patterns belong to different tasks and (b) allocating different networks to learn the different tasks. These issues are addressed in the architecture by combining aspects of competitive learning and associative learning. Specifically, task decompositions are encouraged by enforcing a competition among the networks comprising the architecture. As a result of the competition, different networks learn different training patterns and, thus, learn to compute different functions. The architecture was first presented in Jacobs, Jordan, Nowlan, and Hinton (1991), and combines earlier work on learning task decompositions in a modular architecture by Jacobs, Jordan, and Barto (1991) with the mixture models view of competitive learning advocated by Nowlan (1990) and Hinton and Nowlan (1990). The architecture is also presented elsewhere in this volume by Nowlan and Hinton (1991).

The architecture, which is illustrated in Figure 1, consists of two types of networks: *expert networks* and a *gating network*. The expert networks compete to learn the training patterns and the gating network mediates this competition. Whereas the expert networks have an arbitrary connectivity, the gating network is restricted to have as many output units as there are expert networks, and the activations of these output units must be nonnegative and sum to one. To meet these constraints, we use the "softmax" activation function (Bridle, 1989); specifically, the activation of

the $i^{\text{th}}$ output unit of the gating network, denoted $g_i$, is

$$g_i = \frac{e^{s_i}}{\displaystyle\sum_{j=1}^{n} e^{s_j}} \tag{1}$$

where $s_i$ denotes the weighted sum of unit $i$'s inputs and $n$ denotes the number of expert networks. The output of the entire architecture, denoted $\mathbf{y}$, is

$$\mathbf{y} = \sum_{i=1}^{n} g_i \mathbf{y}_i \tag{2}$$

where $\mathbf{y}_i$ denotes the output of the $i^{\text{th}}$ expert network. During training, the weights of the expert and gating networks are adjusted simultaneously using the backpropagation algorithm (le Cun, 1985; Parker, 1985; Rumelhart, Hinton, and Williams, 1986; Werbos, 1974) so as to maximize the function

$$\ln L = \ln \sum_{i=1}^{n} g_i e^{-\frac{1}{2\sigma_i^2}\|\mathbf{y}^*-\mathbf{y}_i\|^2} \tag{3}$$

where $\mathbf{y}^*$ denotes the target vector and $\sigma_i^2$ denotes a scaling parameter associated with the $i^{\text{th}}$ expert network.

This architecture is best understood if it is given a probabilistic interpretation as an "associative gaussian mixture model" (see Duda and Hart (1973) and McLachlan and Basford (1988) for a discussion of non–associative gaussian mixture models). Under this interpretation, the training patterns are assumed to be generated by a number of different probabilistic rules. At each time step, a rule is selected with probability $g_i$ and a training pattern is generated by the rule. Each rule is characterized by a statistical model of the form $\mathbf{y}^* = f_i(\mathbf{x})+\epsilon_i$, where $f_i(\mathbf{x})$ is a fixed nonlinear function of the input vector, denoted $\mathbf{x}$, and $\epsilon_i$ is a random variable. If it is assumed that $\epsilon_i$ is gaussian with covariance matrix $\sigma_i^2\,\mathbf{I}$, then the residual vector $\mathbf{y}^* - \mathbf{y}_i$ is also gaussian and the cost function in Equation 3 is the log likelihood of generating a particular target vector $\mathbf{y}^*$.

The goal of the architecture is to model the distribution of training patterns. This is achieved by gradient ascent in the log likelihood function. To compute the gradient consider first the partial derivative of the log likelihood with respect to the weighted sum $s_i$ at the $i^{th}$ output unit of the gating network. Using the chain rule and Equation 1 we find that this derivative is given by:

$$\frac{\partial \ln L}{\partial s_i} = g(i \mid \mathbf{x}, \mathbf{y}^*) - g_i \tag{4}$$

where $g(i \mid \mathbf{x}, \mathbf{y}^*)$ is the a posteriori probability that the $i^{th}$ expert network generates the target vector:

$$g(i \mid \mathbf{x}, \mathbf{y}^*) = \frac{g_i e^{-\frac{1}{2\sigma_i^2}\|\mathbf{y}^*-\mathbf{y}_i\|^2}}{\displaystyle\sum_{j=1}^{n} g_j e^{-\frac{1}{2\sigma_j^2}\|\mathbf{y}^*-\mathbf{y}_j\|^2}}. \tag{5}$$

Thus the weights of the gating network are adjusted so that the network's outputs—the a priori probabilities $g_i$—move toward the a posteriori probabilities.

Consider now the gradient of the log likelihood with respect to the output of the $i^{th}$ expert network. Differentiation of $\ln L$ with respect to $\mathbf{y}_i$ yields:

$$\frac{\partial \ln L}{\partial \mathbf{y}_i} = g(i \mid \mathbf{x}, \mathbf{y}^*)\frac{(\mathbf{y}^* - \mathbf{y}_i)}{\sigma_i^2}. \tag{6}$$

These derivatives involve the error term $\mathbf{y}^* - \mathbf{y}_i$ weighted by the a posteriori probability associated with the $i^{th}$ expert network. Thus the weights of the network are adjusted to correct the error between the output of the $i^{th}$ network and the global target vector, but only in proportion to the a posteriori probability. For each input vector, typically only one expert network has a large a posteriori probability. Consequently, only one expert network tends to learn each training pattern. In general, different expert networks learn different training patterns and, thus, learn to compute different functions.

## 3    THE WHAT AND WHERE VISION TASKS

We applied the modular connectionist architecture to the object recognition task ("what" task) and spatial localization task ("where" task) studied by Rueckl, Cave, and Kosslyn (1989).[1] At each time step of the simulation, one of nine objects is placed at one of nine locations on a simulated retina. The "what" task is to identify the object; the "where" task is to identify its location.

The modular architecture is shown in Figure 2. It consists of three expert networks and a gating network. The expert networks receive the retinal image and a task specifier indicating whether the architecture should perform the "what" task or the "where" task at the current time step. The gating network receives the task specifier. The first expert network contains 36 hidden units, the second expert network contains 18 hidden units, and the third expert network doesn't contain any hidden units (i.e., it is a single–layer network).

There are at least three ways that this modular architecture might successfully learn the "what" and "where" tasks. One of the multi–layer expert networks could learn to perform both tasks. Because this solution doesn't show any task decomposition, we consider it to be unsatisfactory. A second possibility is that one of the multi–layer expert networks could learn the "what" task, and the other multi–layer expert network could learn the "where" task. Although this solution exhibits task decomposition, a shortcoming of this solution is apparent when it is noted that, using the retinal images designed by Rueckl et al. (1989), the "where" task is linearly separable. This means that the structure of the single–layer expert network most closely matches the "where" task. Consequently, a third and possibly best solution would be one in which one of the multi–layer expert networks learns the "what" task and the single–layer expert network learns the "where" task. This solution would not only show task decomposition but also the appropriate allocation of tasks to expert networks. Simulation results show that the third possible solution is the one that

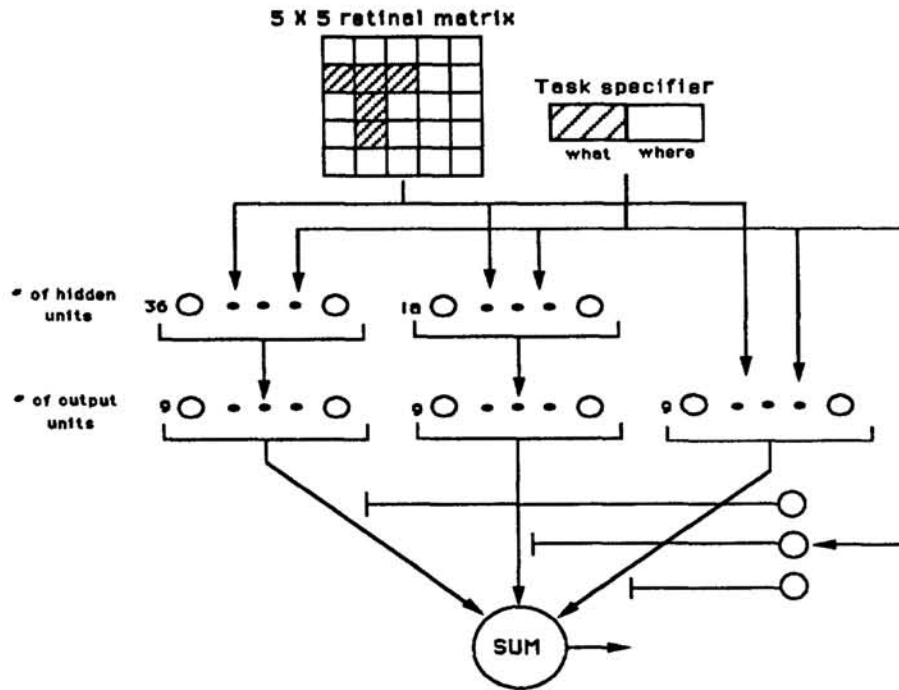

Figure 2: The Modular Architecture Applied to the What and Where Tasks

is always achieved. These results provide evidence that the modular architecture is capable of allocating a different network to different tasks and of allocating a network with an appropriate structure to each task.

## 4    THE MULTI–PAYLOAD ROBOTICS TASK

When designing a compensator for a nonlinear plant, control engineers frequently find it impossible or impractical to design a continuous control law that is useful in all the relevant regions of a plant's parameter space. Typically, the solution to this problem is to use *gain scheduling*; if it is known how the dynamics of a plant change with its operating conditions, then it may be possible to design a piecewise controller that employs different control laws when the plant is operating under different conditions. From our viewpoint, gain scheduling is an attractive solution because it involves task decomposition. It circumvents the problem of determining a fixed global model of the plant dynamics. Instead, the dynamics are approximated using local models that vary with the plant's operating conditions.

Task decomposition is a useful strategy not only when the control law is designed, but also when it is learned. We suggest that an ideal controller is one that, like gain scheduled controllers, uses local models of the plant dynamics, and like learning controllers, learns useful control laws despite uncertainties about the plant or environment. Because the modular connectionist architecture is capable of both task decomposition and learning, it may be useful in achieving both of these desiderata.

We applied the modular architecture to the problem of learning a feedforward con-

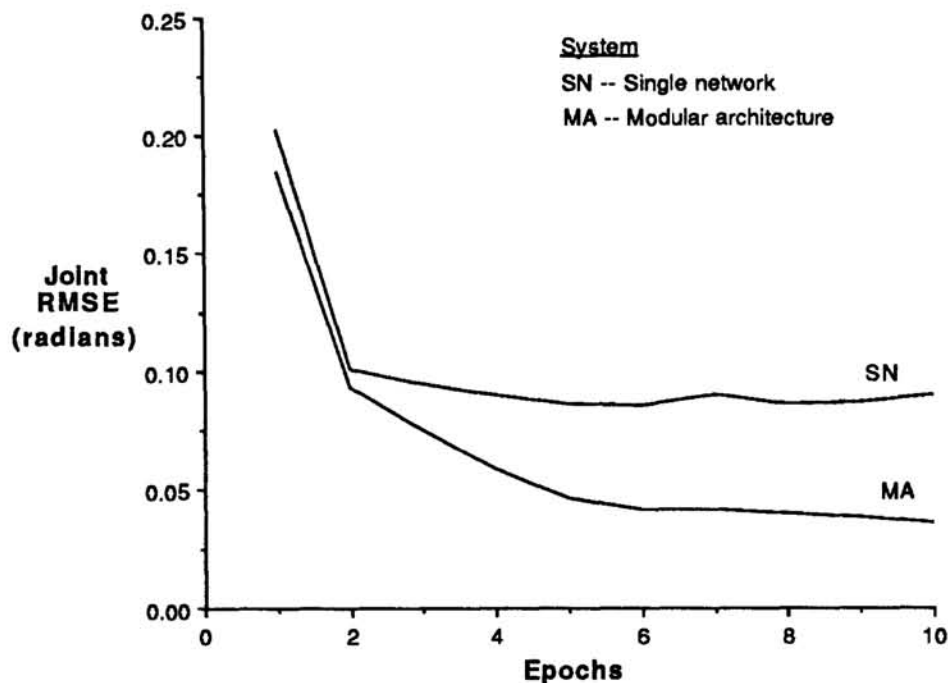

Figure 3: Learning Curves for the Multi–Payload Robotics Task

troller for a robotic arm in a multiple payload task.[2] The task is to drive a simulated two–joint robot arm with a variety of payloads, each of a different mass, along a desired trajectory. The architecture is given the payload's identity (e.g., payload $A$ or payload $B$) but not its mass.

The modular architecture consisted of six expert networks and a gating network. The expert networks received as input the state of the robot arm and the desired acceleration. The gating network received the payload identity. We also trained a single multi–layer network to perform this task. The learning curves for the two systems are shown in Figure 3. The horizontal axis gives the training time in epochs. The vertical axis gives the joint root mean square error in radians. Clearly, the modular architecture learned significantly faster than the single network. Furthermore, the modular architecture learned to perform the task by allocating different expert networks to control the arm with payloads from different mass categories (e.g., light, medium, or heavy payloads).

**Acknowledgements**

This research was supported by a postdoctoral fellowship provided to the first author from the McDonnell–Pew Program in Cognitive Neuroscience, by funding provided to the second author from the Siemens Corporation, and by NSF grant IRI-9013991 awarded to both authors.

## Footnotes

[1] For a detailed presentation of the application of an earlier modular architecture to the "what" and "where" tasks see Jacobs, Jordan, and Barto (1991).

[2]For a detailed presentation of the application of the modular architecture to the multiple payload robotics task see Jacobs and Jordan (1991).

## References

Bridle, J. (1989) Probabilistic interpretation of feedforward classification network outputs, with relationships to statistical pattern recognition. In F. Fogelman-Soulie & J. Hérault (Eds.), *Neuro-computing: Algorithms, Architectures, and Applications.* New York: Springer–Verlag.

Duda, R.O. & Hart, P.E. (1973) *Pattern Classification and Scene Analysis.* New York: John Wiley & Sons.

Hinton, G.E. & Nowlan, S.J. (1990) The bootstrap Widrow–Hoff rule as a cluster-formation algorithm. *Neural Computation,* 2, 355–362.

Jacobs, R.A. & Jordan, M.I. (1991) Learning piecewise control strategies in a modular connectionist architecture. Submitted to *IEEE Transactions on Neural Networks.*

Jacobs, R.A., Jordan, M.I., & Barto, A.G. (1991) Task decomposition through competition in a modular connectionist architecture: The what and where vision tasks. *Cognitive Science,* in press.

Jacobs, R.A., Jordan, M.I., Nowlan, S.J., & Hinton, G.E. (1991) Adaptive mixtures of local experts. *Neural Computation,* in press.

le Cun, Y. (1985) Une procédure d'apprentissage pour réseau a seuil asymétrique [A learning procedure for asymmetric threshold network]. *Proceedings of Cognitiva,* 85, 599–604.

McLachlan, G.J. & Basford, K.E. (1988) *Mixture Models: Inference and Applications to Clustering.* New York: Marcel Dekker.

Nowlan, S.J. (1990) Maximum likelihood competitive learning. In D.S. Touretzky (Ed.), *Advances in Neural Information Processing Systems 2.* San Mateo, CA: Morgan Kaufmann Publishers.

Nowlan, S.J. & Hinton, G.E. (1991) Evaluation of an associative mixture architecture on a vowel recognition task. In R.P. Lippmann, J. Moody, & D.S. Touretzky (Eds.), *Advances in Neural Information Processing Systems 3.* San Mateo, CA: Morgan Kaufmann Publishers.

Parker, D.B. (1985) Learning logic. Technical Report TR–47, Massachusetts Institute of Technology, Cambridge, MA.

Rueckl, J.G., Cave, K.R., & Kosslyn, S.M. (1989) Why are "what" and "where" processed by separate cortical visual systems? A computational investigation. *Journal of Cognitive Neuroscience,* 1, 171–186.

Rumelhart, D.E., Hinton, G.E., & Williams, R.J. (1986) Learning internal representations by error propagation. In D.E. Rumelhart, J.L. McClelland, & the PDP Research Group, *Parallel Distributed Processing: Explorations in the Microstructure of Cognition. Volume 1: Foundations.* Cambridge, MA: The MIT Press.

Werbos, P.J. (1974) *Beyond Regression: New Tools for Prediction and Analysis in the Behavioral Sciences.* Ph.D. thesis, Harvard University, Cambridge, MA.